# Locality-Sensitive Binary Codes from Shift-Invariant Kernels

**Maxim Raginsky**
Duke University
Durham, NC 27708
m.raginsky@duke.edu

**Svetlana Lazebnik**
UNC Chapel Hill
Chapel Hill, NC 27599
lazebnik@cs.unc.edu

## Abstract

This paper addresses the problem of designing binary codes for high-dimensional data such that vectors that are similar in the original space map to similar binary strings. We introduce a simple distribution-free encoding scheme based on random projections, such that the expected Hamming distance between the binary codes of two vectors is related to the value of a shift-invariant kernel (e.g., a Gaussian kernel) between the vectors. We present a full theoretical analysis of the convergence properties of the proposed scheme, and report favorable experimental performance as compared to a recent state-of-the-art method, spectral hashing.

## 1 Introduction

Recently, there has been a lot of interest in the problem of designing compact binary codes for reducing storage requirements and accelerating search and retrieval in large collections of high-dimensional vector data [11, 13, 15]. A desirable property of such coding schemes is that they should map similar data points to similar binary strings, i.e., strings with a low Hamming distance. Hamming distances can be computed very efficiently in hardware, resulting in very fast retrieval of strings similar to a given query, even for brute-force search in a database consisting of millions of data points [11, 13]. Moreover, if code strings can be effectively used as hash keys, then similarity searches can be carried out in sublinear time. In some existing schemes, e.g. [11, 13], the notion of similarity between data points comes from supervisory information, e.g., two documents are similar if they focus on the same topic or two images are similar if they contain the same objects. The binary encoder is then trained to reproduce this "semantic" similarity measure. In this paper, we are more interested in *unsupervised* schemes, where the similarity is given by Euclidean distance or by a kernel defined on the original feature space. Weiss et al. [15] have recently proposed a *spectral hashing* approach motivated by the idea that a good encoding scheme should minimize the sum of Hamming distances between pairs of code strings weighted by the value of a Gaussian kernel between the corresponding feature vectors. With appropriate heuristic simplifications, this objective can be shown to yield a very efficient encoding rule, where each bit of the code is given by the sign of a sine function applied to a one-dimensional projection of the feature vector. Spectral hashing shows promising experimental results, but its behavior is not easy to characterize theoretically. In particular, it is not clear whether the Hamming distance between spectral hashing code strings converges to any function of the Euclidean distance or the kernel value between the original vectors as the number of bits in the code increases.

In this paper, we propose a coding method that is similar to spectral hashing computationally, but is derived from completely different considerations, is amenable to full theoretical analysis, and shows better practical behavior as a function of code size. We start with a low-dimensional mapping of the original data that is guaranteed to preserve the value of a shift-invariant kernel (specifically, the *random Fourier features* of Rahimi and Recht [8]), and convert this mapping to a binary one with similar guarantees. In particular, we show that the *normalized* Hamming distance (i.e., Ham-

ming distance divided by the number of bits in the code) between any two embedded points sharply concentrates around a well-defined continuous function of the kernel value. This leads to a Johnson–Lindenstrauss type result [4] which says that a set of any $N$ points in a Euclidean feature space can be embedded in a binary cube of dimension $O(\log N)$ in a similarity-preserving way: with high probability, the binary encodings of any two points that are similar (as measured by the kernel) are nearly identical, while those of any two points that are dissimilar differ in a constant fraction of their bits. Using entropy bounds from the theory of empirical processes, we also prove a stronger result of this type that holds for any *compact* domain of $\mathbb{R}^D$, provided the number of bits is proportional to the *intrinsic dimension* of the domain. Our scheme is completely distribution-free with respect to the data: its structure depends only on the underlying kernel. In this, it is similar to *locality sensitive hashing* (LSH) [1], which is a family of methods for deriving low-dimensional discrete representations of the data for sublinear near-neighbor search. However, our scheme differs from LSH in that we obtain both upper and lower bounds on the normalized Hamming distance between any two embedded points, while in LSH the goal is only to preserve nearest neighbors (see [6] for further discussion of the distinction between LSH and more general similarity-preserving embeddings). To the best of our knowledge, our scheme is among the first random projection methods for constructing a similarity-preserving embedding into a binary cube. In addition to presenting a thorough theoretical analysis, we have evaluated our approach on both synthetic and real data (images from the LabelMe database [10] represented by high-dimensional GIST descriptors [7]) and compared its performance to that of spectral hashing. Despite the simplicity and distribution-free nature of our scheme, we have been able to obtain very encouraging experimental results.

## 2 Binary codes for shift-invariant kernels

Consider a Mercer kernel $K(\cdot, \cdot)$ on $\mathbb{R}^D$ that satisfies the following for all points $\boldsymbol{x}, \boldsymbol{y} \in \mathbb{R}^D$:

(K1) It is *translation-invariant* (or *shift-invariant*), i.e., $K(\boldsymbol{x}, \boldsymbol{y}) = K(\boldsymbol{x} - \boldsymbol{y})$.

(K2) It is *normalized*, i.e., $K(\boldsymbol{x} - \boldsymbol{y}) \leq 1$ and $K(\boldsymbol{x} - \boldsymbol{x}) \equiv K(\boldsymbol{0}) = 1$.

(K3) For any real number $\alpha \geq 1$, $K(\alpha\boldsymbol{x} - \alpha\boldsymbol{y}) \leq K(\boldsymbol{x} - \boldsymbol{y})$.

The Gaussian kernel $K(\boldsymbol{x}, \boldsymbol{y}) = \exp(-\gamma\|\boldsymbol{x} - \boldsymbol{y}\|^2/2)$ or the Laplacian kernel $K(\boldsymbol{x}, \boldsymbol{y}) = \exp(-\gamma\|\boldsymbol{x} - \boldsymbol{y}\|_1)$ are two well-known examples. We would like to construct an embedding $F^n$ of $\mathbb{R}^D$ into the binary cube $\{0, 1\}^n$ such that for any pair $\boldsymbol{x}, \boldsymbol{y}$ the normalized Hamming distance

$$\frac{1}{n} d_H(F^n(\boldsymbol{x}), F^n(\boldsymbol{y})) \triangleq \frac{1}{n} \sum_{i=1}^{n} 1_{\{F_i(\boldsymbol{x}) \neq F_i(\boldsymbol{y})\}}$$

between $F^n(\boldsymbol{x}) = (F_1(\boldsymbol{x}), \ldots, F_n(\boldsymbol{x}))$ and $F^n(\boldsymbol{y}) = (F_1(\boldsymbol{y}), \ldots, F_n(\boldsymbol{y}))$ behaves like

$$h_1(K(\boldsymbol{x} - \boldsymbol{y})) \leq \frac{1}{n} d_H(F^n(\boldsymbol{x}), F^n(\boldsymbol{y})) \leq h_2(K(\boldsymbol{x} - \boldsymbol{y}))$$

where $h_1, h_2 : [0, 1] \to \mathbb{R}^+$ are continuous decreasing functions, and $h_1(1) = h_2(1) = 0$ and $h_1(0) = h_2(0) = c > 0$. In other words, we would like to map $D$-dimensional real vectors into $n$-bit binary strings in a locality-sensitive manner, where the notion of locality is induced by the kernel $K$. We will achieve this goal by drawing $F^n$ appropriately at random.

**Random Fourier features.** Recently, Rahimi and Recht [8] gave a scheme that takes a Mercer kernel satisfying (K1) and (K2) and produces a *random* mapping $\Phi^n : \mathbb{R}^D \to \mathbb{R}^n$, such that, with high probability, the inner product of any two transformed points approximates the kernel: $\Phi^n(\boldsymbol{x}) \cdot \Phi^n(\boldsymbol{y}) \approx K(\boldsymbol{x} - \boldsymbol{y})$ for all $\boldsymbol{x}, \boldsymbol{y}$. Their scheme exploits Bochner's theorem [9], a fundamental result in harmonic analysis which says that any such $K$ is a Fourier transform of a uniquely defined probability measure $P_K$ on $\mathbb{R}^D$. They define the *random Fourier features* (RFF) via

$$\Phi_{\boldsymbol{\omega}, b}(\boldsymbol{x}) \triangleq \sqrt{2} \cos(\boldsymbol{\omega} \cdot \boldsymbol{x} + b), \tag{1}$$

where $\boldsymbol{\omega} \sim P_K$ and $b \sim \text{Unif}[0, 2\pi]$. For example, for the Gaussian kernel $K(\boldsymbol{s}) = e^{-\gamma\|\boldsymbol{s}\|^2/2}$, we take $\boldsymbol{\omega} \sim \text{Normal}(0, \gamma I_{D \times D})$. With these features, we have $\mathbb{E}[\Phi_{\boldsymbol{\omega}, b}(\boldsymbol{x})\Phi_{\boldsymbol{\omega}, b}(\boldsymbol{y})] = K(\boldsymbol{x} - \boldsymbol{y})$. The scheme of [8] is as follows: draw an i.i.d. sample $((\boldsymbol{\omega}_1, b_1), \ldots, (\boldsymbol{\omega}_n, b_n))$, where each

$\boldsymbol{\omega}_i \sim P_K$ and $b_i \sim \text{Unif}[0, 2\pi]$, and define a mapping $\Phi^n : \mathbb{R}^D \to \mathbb{R}^n$ via $\Phi^n(\boldsymbol{x}) \triangleq \frac{1}{\sqrt{n}}(\Phi_{\boldsymbol{\omega}_1, b_1}(\boldsymbol{x}), \ldots, \Phi_{\boldsymbol{\omega}_n, b_n}(\boldsymbol{x}))$ for $\boldsymbol{x} \in \mathcal{X}$. Then $\mathbb{E}[\Phi^n(\boldsymbol{x}) \cdot \Phi^n(\boldsymbol{y})] = K(\boldsymbol{x} - \boldsymbol{y})$ for all $\boldsymbol{x}, \boldsymbol{y}$.

**From random Fourier features to random binary codes.** We will compose the RFFs with *random binary quantizers*. Draw a random *threshold* $t \sim \text{Unif}[-1, 1]$ and define the quantizer $Q_t : [-1, 1] \to \{-1, +1\}$ via $Q_t(u) \triangleq \text{sgn}(u + t)$, where we let $\text{sgn}(u) = -1$ if $u < 0$ and $\text{sgn}(u) = +1$ if $u \geq 0$. We note the following basic fact (we omit the easy proof):

**Lemma 2.1** *For any $u, v \in [-1, 1]$, $\mathbb{P}_t\{Q_t(u) \neq Q_t(v)\} = |u - v|/2$.*

Now, given a kernel $K$, we define a random map $F_{t,\boldsymbol{\omega},b} : \mathbb{R}^D \to \{0, 1\}$ through

$$F_{t,\boldsymbol{\omega},b}(\boldsymbol{x}) \triangleq \frac{1}{2}\left[1 + Q_t\left(\cos(\boldsymbol{\omega} \cdot \boldsymbol{x} + b)\right)\right], \tag{2}$$

where $t \sim \text{Unif}[-1, 1]$, $\boldsymbol{\omega} \sim P_K$, and $b \sim \text{Unif}[0, 2\pi]$ are independent of one another. From now on, we will often omit the subscripts $t, \boldsymbol{\omega}, b$ and just write $F$ for the sake of brevity. We have:

**Lemma 2.2**

$$\mathbb{E}\,1_{\{F(\boldsymbol{x}) \neq F(\boldsymbol{y})\}} = h_K(\boldsymbol{x} - \boldsymbol{y}) \triangleq \frac{8}{\pi^2} \sum_{m=0}^{\infty} \frac{1 - K(m\boldsymbol{x} - m\boldsymbol{y})}{4m^2 - 1}, \qquad \forall \boldsymbol{x}, \boldsymbol{y} \tag{3}$$

**Proof:** Using Lemma 2.1, we can show $\mathbb{E}\,1_{\{F(\boldsymbol{x}) \neq F(\boldsymbol{y})\}} = \frac{1}{2}\mathbb{E}_{\boldsymbol{\omega},b}|\cos(\boldsymbol{\omega} \cdot \boldsymbol{x} + b) - \cos(\boldsymbol{\omega} \cdot \boldsymbol{y} + b)|$. Using trigonometric identities and the independence of $\boldsymbol{\omega}$ and $b$, we can express this expectation as

$$\mathbb{E}_{b,\boldsymbol{\omega}}|\cos(\boldsymbol{\omega} \cdot \boldsymbol{x} + b) - \cos(\boldsymbol{\omega} \cdot \boldsymbol{y} + b)| = \frac{4}{\pi}\mathbb{E}_{\boldsymbol{\omega}}\left|\sin\left(\frac{\boldsymbol{\omega} \cdot (\boldsymbol{x} - \boldsymbol{y})}{2}\right)\right|.$$

We now make use of the Fourier series representation of the full rectified sine wave $g(\tau) = |\sin(\tau)|$:

$$g(\tau) = \frac{2}{\pi} + \frac{4}{\pi}\sum_{m=1}^{\infty}\frac{1}{1 - 4m^2}\cos(m\tau) = \frac{4}{\pi}\sum_{m=1}^{\infty}\frac{1 - \cos(2m\tau)}{4m^2 - 1}.$$

Using this together with the fact that $\mathbb{E}_{\boldsymbol{\omega}}\cos(\boldsymbol{\omega} \cdot \boldsymbol{s}) = K(\boldsymbol{s})$ for any $\boldsymbol{s} \in \mathbb{R}^D$ [8], we obtain (3). $\blacksquare$

Lemma 2.2 shows that the probability that $F(\boldsymbol{x}) \neq F(\boldsymbol{y})$ is a well-defined continuous function of $\boldsymbol{x} - \boldsymbol{y}$. The infinite series in (3) can, of course, be computed numerically to any desired precision. In addition, we have the following upper and lower bounds solely in terms of the kernel value $K(\boldsymbol{x} - \boldsymbol{y})$:

**Lemma 2.3** *Define the functions*

$$h_1(u) \triangleq \frac{4}{\pi^2}(1 - u) \qquad \text{and} \qquad h_2(u) \triangleq \min\left\{\frac{1}{2}\sqrt{1 - u}, \frac{4}{\pi^2}(1 - 2u/3)\right\},$$

*where $u \in [0, 1]$. Note that $h_1(0) = h_2(0) = 4/\pi^2 \approx 0.405$ and that $h_1(1) = h_2(1) = 0$. Then $h_1(K(\boldsymbol{x} - \boldsymbol{y})) \leq h_K(\boldsymbol{x} - \boldsymbol{y}) \leq h_2(K(\boldsymbol{x} - \boldsymbol{y}))$ for all $\boldsymbol{x}, \boldsymbol{y}$.*

**Proof:** Let $\Delta \triangleq \cos(\boldsymbol{\omega} \cdot \boldsymbol{x} + b) - \cos(\boldsymbol{\omega} \cdot \boldsymbol{y} + b)$. Then $\mathbb{E}|\Delta| = \mathbb{E}\sqrt{\Delta^2} \leq \sqrt{\mathbb{E}\,\Delta^2}$ (the last step uses concavity of the square root). Using the properties of the RFF, $\mathbb{E}\,\Delta^2 = (1/2)\mathbb{E}[(\Phi_{\boldsymbol{\omega},b}(\boldsymbol{x}) - \Phi_{\boldsymbol{\omega},b}(\boldsymbol{y}))^2] = 1 - K(\boldsymbol{x} - \boldsymbol{y})$. Therefore, $\mathbb{E}\,1_{\{F(\boldsymbol{x}) \neq F(\boldsymbol{y})\}} = (1/2)\mathbb{E}|\Delta| \leq (1/2)\sqrt{1 - K(\boldsymbol{x} - \boldsymbol{y})}$. We also have

$$\mathbb{E}\,1_{\{F(\boldsymbol{x}) \neq F(\boldsymbol{y})\}} = \frac{4}{\pi^2} - \frac{8}{\pi^2}\sum_{m=1}^{\infty}\frac{K(m\boldsymbol{x} - m\boldsymbol{y})}{4m^2 - 1} \leq \frac{4}{\pi^2} - \frac{8}{3\pi^2}K(\boldsymbol{x} - \boldsymbol{y}) = \frac{4}{\pi^2}(1 - 2K(\boldsymbol{x} - \boldsymbol{y})/3).$$

This proves the upper bound in the lemma. On the other hand, since $K$ satisfies (K3),

$$h_K(\boldsymbol{x} - \boldsymbol{y}) \geq (1 - K(\boldsymbol{x} - \boldsymbol{y})) \cdot \frac{8}{\pi^2}\sum_{m=1}^{\infty}\frac{1}{4m^2 - 1} = \frac{4}{\pi^2}(1 - K(\boldsymbol{x} - \boldsymbol{y})),$$

because the $m$th term of the series in (3) is not smaller than $(1 - K(\boldsymbol{x} - \boldsymbol{y}))/(4m^2 - 1)$. $\blacksquare$

Fig. 1 shows a comparison of the kernel approximation properties of the RFFs [8] with our scheme for the Gaussian kernel.

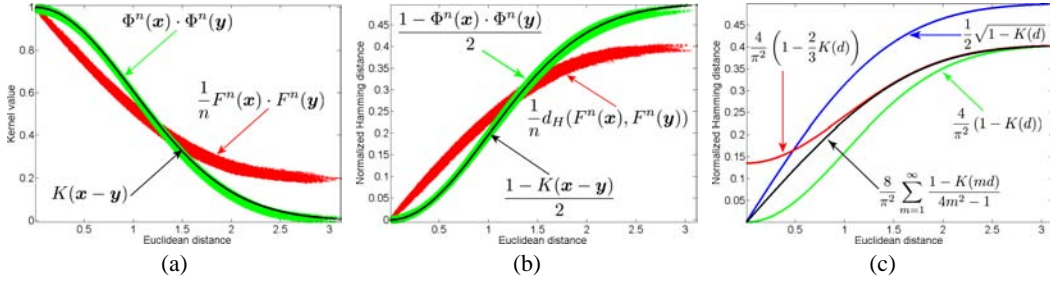

Figure 1: (a) Approximating the Gaussian kernel by random features (green) and random signs (red). (b) Relationship of normalized Hamming distance between random signs to functions of the kernel. The scatter plots in (a) and (b) are obtained from a synthetic set of 500 uniformly distributed 2D points with $n = 5000$. (c) Bounds for normalized Hamming distance in Lemmas 2.2 and 2.3 vs. the Euclidean distance.

Now we concatenate several mappings of the form $F_{t,\boldsymbol{\omega},b}$ to construct an embedding of $\mathcal{X}$ into the binary cube $\{0,1\}^n$. Specifically, we draw $n$ i.i.d. triples $(t_1, \boldsymbol{\omega}_1, b_1), \ldots, (t_n, \boldsymbol{\omega}_n, b_n)$ and define

$$F^n(\boldsymbol{x}) \triangleq \big(F_1(\boldsymbol{x}), \ldots, F_n(\boldsymbol{y})\big), \qquad \text{where } F_i(\boldsymbol{x}) \equiv F_{t_i, \boldsymbol{\omega}_i, b_i}(\boldsymbol{x}), i = 1, \ldots, n$$

As we will show next, this construction ensures that, for any two points $\boldsymbol{x}$ and $\boldsymbol{y}$, the fraction of the bits where the binary strings $F^n(\boldsymbol{x})$ and $F^n(\boldsymbol{y})$ disagree sharply concentrates around $h_K(\boldsymbol{x} - \boldsymbol{y})$, provided $n$ is large enough. Using the results proved above, we conclude that, for any two points $\boldsymbol{x}$ and $\boldsymbol{y}$ that are "similar," i.e., $K(\boldsymbol{x} - \boldsymbol{y}) \sim 1$, most of the bits of $F^n(\boldsymbol{x})$ and $F^n(\boldsymbol{y})$ will agree, whereas for any two points $\boldsymbol{x}$ and $\boldsymbol{y}$ that are "dissimilar," i.e., $K(\boldsymbol{x} - \boldsymbol{y}) \sim 0$, $F^n(\boldsymbol{x})$ and $F^n(\boldsymbol{y})$ will disagree in about $40\%$ or more of their bits.

**Analysis of performance.** We first prove a Johnson–Lindenstrauss type result which says that, for any finite subset of $\mathbb{R}^D$, the normalized Hamming distance respects the similarities between points. It should be pointed out that the analogy with Johnson–Lindenstrauss is only qualitative: our embedding is highly nonlinear, in contrast to random linear projections used there [4], and the resulting distortion of the neighborhood structure, although controllable, does not amount to a mere rescaling by constants.

**Theorem 2.4** *Fix $\epsilon, \delta \in (0, 1)$. For any finite data set $\mathcal{D} = \{\boldsymbol{x}_1, \ldots, \boldsymbol{x}_N\} \subset \mathbb{R}^D$, $F^n$ is such that*

$$h_K(\boldsymbol{x}_j - \boldsymbol{x}_k) - \delta \leq \frac{1}{n} d_H(F^n(\boldsymbol{x}_j), F^n(\boldsymbol{x}_k)) \leq h_K(\boldsymbol{x}_j - \boldsymbol{x}_k) + \delta \tag{4}$$

$$h_1(K(\boldsymbol{x}_j - \boldsymbol{x}_k)) - \delta \leq \frac{1}{n} d_H(F^n(\boldsymbol{x}_j), F^n(\boldsymbol{x}_k)) \leq h_2(K(\boldsymbol{x}_j - \boldsymbol{x}_k)) + \delta \tag{5}$$

*for all $j, k$ with probability $\geq 1 - N^2 e^{-2n\delta^2}$. Moreover, the events (4) and (5) will hold with probability $\geq 1 - \epsilon$ if $n \geq (1/2\delta^2) \log(N^2/\epsilon)$. Thus, any $N$-point subset of $\mathbb{R}^D$ can be embedded, with high probability, into the binary cube of dimension $O(\log N)$ in a similarity-preserving way.*

The proof (omitted) is by a standard argument using Hoeffding's inequality and the union bound, as well as the bounds of Lemma 2.3. We also prove a much stronger result: any compact subset $\mathcal{X} \subset \mathbb{R}^D$ can be embedded into a binary cube whose dimension depends only on the intrinsic dimension and the diameter of $\mathcal{X}$ and on the second moment of $P_K$, such that the normalized Hamming distance behaves in a similarity-preserving way for all pairs of points in $\mathcal{X}$ *simultaneously*. We make use of the following [5]:

**Definition 2.5** *The* Assouad dimension *of $\mathcal{X} \subset \mathbb{R}^D$, denoted by $d_{\mathcal{X}}$, is the smallest integer $k$, such that, for any ball $B \subset \mathbb{R}^D$, the set $B \cap \mathcal{X}$ can be covered by $2^k$ balls of half the radius of $B$.*

The Assouad dimension is a widely used measure of the intrinsic dimension [2, 6, 3]. For example, if $\mathcal{X}$ is an $\ell_p$ ball in $\mathbb{R}^D$, then $d_{\mathcal{X}} = O(D)$; if $\mathcal{X}$ is a $d$-dimensional hyperplane in $\mathbb{R}^D$, then $d_{\mathcal{X}} = O(d)$ [2]. Moreover, if $\mathcal{X}$ is a $d$-dimensional Riemannian submanifold of $\mathbb{R}^D$ with a suitably bounded curvature, then $d_{\mathcal{X}} = O(d)$ [3]. We now have the following result:

**Theorem 2.6** *Suppose that the kernel $K$ is such that $L_K \triangleq \sqrt{\mathbb{E}_{\boldsymbol{\omega} \sim P_K} \|\boldsymbol{\omega}\|^2} < +\infty$. Then there exists a constant $C > 0$ independent of $D$ and $K$, such that the following holds. Fix any $\epsilon, \delta > 0$. If*

$$n \geq \max \left\{ \frac{C L_K d_{\mathcal{X}} \operatorname{diam} \mathcal{X}}{\delta^2}, \frac{2}{\delta^2} \log \left( \frac{2}{\epsilon} \right) \right\},$$

*then, with probability at least $1 - \epsilon$, the mapping $F^n$ is such that, for every pair $\boldsymbol{x}, \boldsymbol{y} \in \mathcal{X}$,*

$$h_K(\boldsymbol{x} - \boldsymbol{y}) - \delta \leq \frac{1}{n} d_H(F^n(\boldsymbol{x}), F^n(\boldsymbol{y})) \leq h_K(\boldsymbol{x} - \boldsymbol{y}) + \delta \tag{6}$$

**Proof:** For every pair $\boldsymbol{x}, \boldsymbol{y} \in \mathcal{X}$, let $A_{\boldsymbol{x}, \boldsymbol{y}}$ be the set of all $\boldsymbol{\theta} \equiv (t, \boldsymbol{\omega}, b)$, such that $F_{t, \boldsymbol{\omega}, b}(\boldsymbol{x}) \neq F_{t, \boldsymbol{\omega}, b}(\boldsymbol{y})$, and let $\mathcal{A} = \{A_{\boldsymbol{x}, \boldsymbol{y}} : \boldsymbol{x}, \boldsymbol{y} \in \mathcal{X}\}$. Then we can write

$$\frac{1}{n} d_H(F^n(\boldsymbol{x}), F^n(\boldsymbol{y})) = \frac{1}{n} \sum_{i=1}^n 1_{\{\boldsymbol{\theta}_i \in A_{\boldsymbol{x}, \boldsymbol{y}}\}}.$$

For any sequence $\boldsymbol{\theta}^n = (\boldsymbol{\theta}_1, \ldots, \boldsymbol{\theta}_n)$, define the uniform deviation

$$\Delta(\boldsymbol{\theta}^n) \triangleq \sup_{\boldsymbol{x}, \boldsymbol{y} \in \mathcal{X}} \left| \frac{1}{n} \sum_{i=1}^n 1_{\{\boldsymbol{\theta}_i \in A_{\boldsymbol{x}, \boldsymbol{y}}\}} - \mathbb{E} \, 1_{\{F_{t, \boldsymbol{\omega}, b}(\boldsymbol{x}) \neq F_{t, \boldsymbol{\omega}, b}(\boldsymbol{y})\}} \right|.$$

For every $1 \leq i \leq n$ and an arbitrary $\boldsymbol{\theta}'_i$, let $\boldsymbol{\theta}^n_{(i)}$ denote $\boldsymbol{\theta}^n$ with the $i$th component replaced by $\boldsymbol{\theta}'_i$. Then $|\Delta(\boldsymbol{\theta}^n) - \Delta(\boldsymbol{\theta}^n_{(i)})| \leq 1/n$ for any $i$ and any $\boldsymbol{\theta}'_i$. Hence, by McDiarmid's inequality,

$$\mathbb{P} \{ |\Delta(\boldsymbol{\theta}^n) - \mathbb{E}_{\boldsymbol{\theta}^n} \Delta(\boldsymbol{\theta}^n)| > \beta \} \leq 2 e^{-2n\beta^2}, \qquad \forall \beta > 0. \tag{7}$$

Now we need to bound $\mathbb{E}_{\boldsymbol{\theta}^n} \Delta(\boldsymbol{\theta}^n)$. Using a standard symmetrization technique [14], we can write

$$\mathbb{E}_{\boldsymbol{\theta}^n} \Delta(\boldsymbol{\theta}^n) \leq 2R(\mathcal{A}) \triangleq 2 \, \mathbb{E}_{\boldsymbol{\theta}^n, \sigma^n} \left[ \sup_{\boldsymbol{x}, \boldsymbol{y} \in \mathcal{X}} \left| \frac{1}{n} \sum_{i=1}^n \sigma_i 1_{\{\boldsymbol{\theta}_i \in A_{\boldsymbol{x}, \boldsymbol{y}}\}} \right| \right], \tag{8}$$

where $\sigma^n = (\sigma_1, \ldots, \sigma_n)$ is an i.i.d. Rademacher sequence, $\mathbb{P}\{\sigma_i = -1\} = \mathbb{P}(\sigma_i = +1) = 1/2$. The quantity $R(\mathcal{A})$ can be bounded by the Dudley entropy integral [14]

$$R(\mathcal{A}) \leq \frac{C_0}{\sqrt{n}} \int_0^\infty \sqrt{\log N(\epsilon, \mathcal{A}, \| \cdot \|_{L^2(\mu)})} d\epsilon, \tag{9}$$

where $C_0 > 0$ is a universal constant, and $N(\epsilon, \mathcal{A}, \| \cdot \|_{L^2(\mu)})$ is the $\epsilon$-covering number of the function class $\{\boldsymbol{\theta} \mapsto 1_{\{\boldsymbol{\theta} \in A\}} : A \in \mathcal{A}\}$ with respect to the $L^2(\mu)$ norm, where $\mu$ is the distribution of $\boldsymbol{\theta} \equiv (t, \boldsymbol{\omega}, b)$. We will bound these covering numbers by the covering numbers of $\mathcal{X}$ with respect to the Euclidean norm on $\mathbb{R}^D$. It can be shown that, for any four points $\boldsymbol{x}, \boldsymbol{x}', \boldsymbol{y}, \boldsymbol{y}' \in \mathcal{X}$,

$$\left\| 1_{A_{\boldsymbol{x}, \boldsymbol{y}}} - 1_{A_{\boldsymbol{x}', \boldsymbol{y}'}} \right\|_{L^2(\mu)}^2 = \int \left( 1_{\{\boldsymbol{\theta} \in A_{\boldsymbol{x}, \boldsymbol{y}}\}} - 1_{\{\boldsymbol{\theta} \in A_{\boldsymbol{x}', \boldsymbol{y}'}\}} \right)^2 d\mu(\boldsymbol{\theta}) \leq \mu(B_{\boldsymbol{x}} \triangle B_{\boldsymbol{x}'}) + \mu(B_{\boldsymbol{y}} \triangle B_{\boldsymbol{y}'}),$$

where $\triangle$ denotes symmetric difference of sets, and $B_{\boldsymbol{x}} \triangleq \{(t, \boldsymbol{\omega}, b) : Q_t(\cos(\boldsymbol{\omega} \cdot \boldsymbol{x} + b)) = +1\}$ (details omitted for lack of space). Now,

$$
\begin{aligned}
2\mu(B_{\boldsymbol{x}} \triangle B_{\boldsymbol{x}'}) &= 2 \, \mathbb{E}_{\boldsymbol{\omega}, b} \left[ \mathbb{P}_t \left\{ Q_t(\cos(\boldsymbol{\omega} \cdot \boldsymbol{x} + b)) \neq Q_t(\cos(\boldsymbol{\omega} \cdot \boldsymbol{y} + b)) \right\} \right] \\
&= \mathbb{E}_{\boldsymbol{\omega}, b} |\cos(\boldsymbol{\omega} \cdot \boldsymbol{x} + b) - \cos(\boldsymbol{\omega} \cdot \boldsymbol{x}' + b)| \leq \mathbb{E}_{\boldsymbol{\omega}} |\boldsymbol{\omega} \cdot (\boldsymbol{x} - \boldsymbol{x}')| \leq L_K \|\boldsymbol{x} - \boldsymbol{x}'\|.
\end{aligned}
$$

Then $\mu(B_{\boldsymbol{x}} \triangle B_{\boldsymbol{x}'}) + \mu(B_{\boldsymbol{y}} \triangle B_{\boldsymbol{y}'}) \leq \frac{L_K}{2} (\|\boldsymbol{x} - \boldsymbol{x}'\| + \|\boldsymbol{y} - \boldsymbol{y}'\|)$. This implies that $N(\epsilon, \mathcal{A}, \| \cdot \|_{L^2(\mu)}) \leq N(\epsilon^2/L_K, \mathcal{X}, \|\cdot\|)^2$, where $N(\delta, \mathcal{X}, \|\cdot\|)$ are the covering numbers of $\mathcal{X}$ w.r.t. the Euclidean norm $\|\cdot\|$. By definition of the Assouad dimension, $N(\delta, \mathcal{X}, \|\cdot\|) \leq (2 \operatorname{diam} \mathcal{X}/\delta)^{d_{\mathcal{X}}}$, so $N(\epsilon, \mathcal{A}, \| \cdot \|_{L^2(\mu)}) \leq \left( \frac{2 L_K \operatorname{diam} \mathcal{X}}{\epsilon^2} \right)^{2d_{\mathcal{X}}}$. We can now estimate the integral in (9) by

$$R(\mathcal{A}) \leq C_1 \sqrt{\frac{L_K d_{\mathcal{X}} \operatorname{diam} \mathcal{X}}{n}}, \tag{10}$$

for some constant $C_1 > 0$. From (10) and (8), we obtain $\mathbb{E}_{\boldsymbol{\theta}^n} \Delta(\boldsymbol{\theta}^n) \leq C_2 \sqrt{\frac{L_K d_{\mathcal{X}} \operatorname{diam} \mathcal{X}}{n}}$, where $C_2 = 2C_1$. Using this and (7) with $\beta = \delta/2$, we obtain (6) with $C = 16C_2^2$. $\blacksquare$

For example, with the Gaussian kernel $K(\boldsymbol{s}) = e^{-\gamma \|\boldsymbol{s}\|^2/2}$ on $\mathbb{R}^D$, we have $L_K = \sqrt{D\gamma}$. The kernel bandwidth $\gamma$ is often chosen as $\gamma \propto 1/[D(\operatorname{diam} \mathcal{X})^2]$ (see, e.g., [12, Sec. 7.8]); with this setting, the number of bits needed to guarantee the bound (6) is $n = \Omega((d_{\mathcal{X}}/\delta^2) \log(1/\epsilon))$. It is possible, in principle, to construct a *dimension-reducing* embedding of $\mathcal{X}$ into a binary cube, provided the number of bits in the embedding is larger than the intrinsic dimension of $\mathcal{X}$.

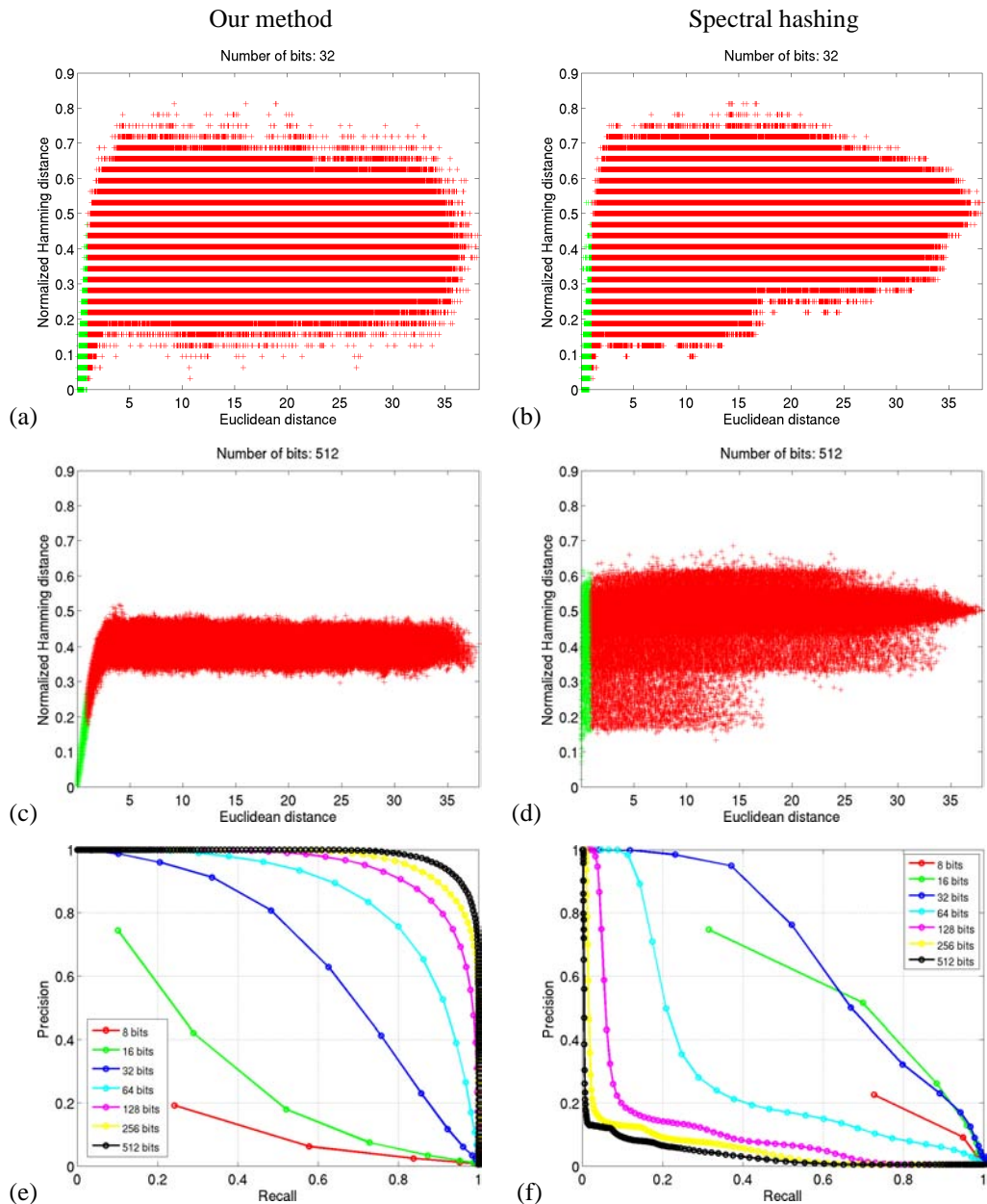

Figure 2: Synthetic results. First row: scatter plots of normalized Hamming distance vs. Euclidean distance for our method (a) and spectral hashing (b) with code size 32 bits. Green indicates pairs of data points that are considered true "neighbors" for the purpose of retrieval. Second row: scatter plots for our method (c) and spectral hashing (d) with code size 512 bits. Third row: recall-precision plots for our method (e) and spectral hashing (f) for code sizes from 8 to 512 bits (best viewed in color).

## 3   Empirical Evaluation

In this section, we present the results of our scheme with a Gaussian kernel, and compare our performance to spectral hashing [15].[1] Spectral hashing is a recently introduced, state-of-the-art approach that has been reported to obtain better results than several other well-known methods, including LSH [1] and restricted Boltzmann machines [11]. Unlike our method, spectral hashing chooses code parameters in a deterministic, data-dependent way, motivated by results on convergence of

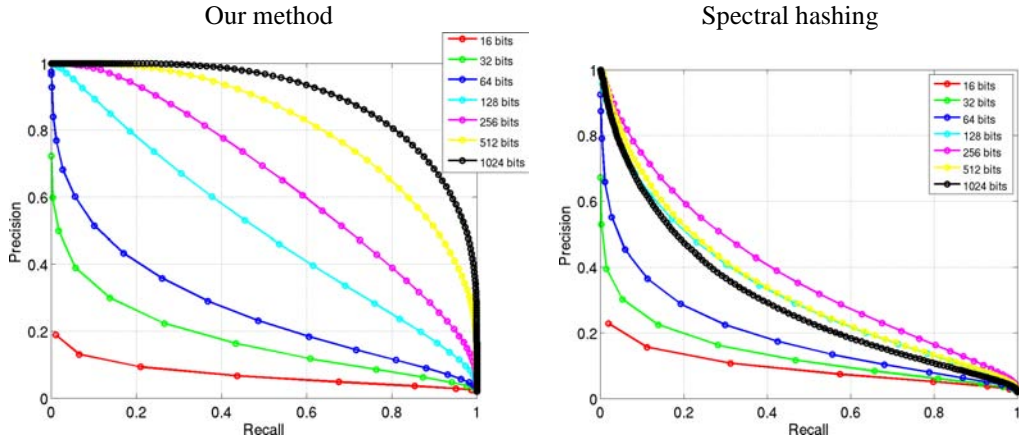

Figure 3: Recall-precision curves for the LabelMe database for our method (left) and for spectral hashing (right). Best viewed in color.

eigenvectors of graph Laplacians to Laplacian eigenfunctions on manifolds. Though spectral hashing is derived from completely different considerations than our method, its encoding scheme is similar to ours in terms of basic computation. Namely, each bit of a spectral hashing code is given by $\mathrm{sgn}(\cos(k\,\boldsymbol{\omega}\cdot\boldsymbol{x}))$, where $\boldsymbol{\omega}$ is a principal direction of the data (instead of a randomly sampled direction, as in our method) and $k$ is a weight that is deterministically chosen according to the analytical form of certain kinds of Laplacian eigenfunctions. The structural similarity between spectral hashing and our method makes comparison between them appropriate.

To demonstrate the basic behavior of our method, we first report results for two-dimensional synthetic data using a protocol similar to [15] (we have also conducted tests on higher-dimensional synthetic data, with very similar results). We sample 10,000 "database" and 1,000 "query" points from a uniform distribution defined on a 2d rectangle with aspect ratio 0.5. To distinguish true positives from false positives for evaluating retrieval performance, we select a "nominal" neighborhood radius so that each query point on average has 50 neighbors in the database. Next, we rescale the data so that this radius is 1, and set the bandwidth of the kernel to $\gamma = 1$. Fig. 2 (a,c) shows scatter plots of normalized Hamming distance vs. Euclidean distance for each query point paired with each database point for 32-bit and 512-bit codes. As more bits are added to our code, the variance of the scatter plots decreases, and the points cluster tighter around the theoretically expected curve (Eq. (3), Fig. 1). The scatter plots for spectral hashing are shown in Fig. 2 (b,d). As the number of bits in the spectral hashing code is increased, normalized Hamming distance does not appear to converge to any clear function of the Euclidean distance. Because the derivation of spectral hashing in [15] includes several heuristic steps, the behavior of the resulting scheme appears to be difficult to analyze, and shows some undesirable effects as the code size increases. Figure 2 (e,f) compares recall-precision curves for both methods using a range of code sizes. Since the normalized Hamming distance for our method converges to a monotonic function of the Euclidean distance, its performance keeps improving as a function of code size. On the other hand, spectral hashing starts out with promising performance for very short codes (up to 32 bits), but then deteriorates for higher numbers of bits.

Next, we present retrieval results for 14,871 images taken from the LabelMe database [10]. The images are represented by 320-dimensional GIST descriptors [7], which have proven to be effective at capturing perceptual similarity between scenes. For this experiment, we randomly select 1,000 images to serve as queries, and the rest make up the "database." As with the synthetic experiments, a nominal threshold of the average distance to the 50th nearest neighbor is used to determine whether a database point returned for a given query is considered a true positive. Figure 3 shows precision-recall curves for code sizes ranging from 16 bits to 1024 bits. As in the synthetic experiments, spectral hashing appears to have an advantage over our method for extremely small code sizes, up to about 32 bits. However, this low bit regime may not be very useful in practice, since below 32 bits, neither method achieves performance levels that would be satisfactory for real-world applications. For larger code sizes, our method begins to dominate. For example, with a 128-bit code (which is equivalent to just two double-precision floating point numbers), our scheme achieves 0.8 precision

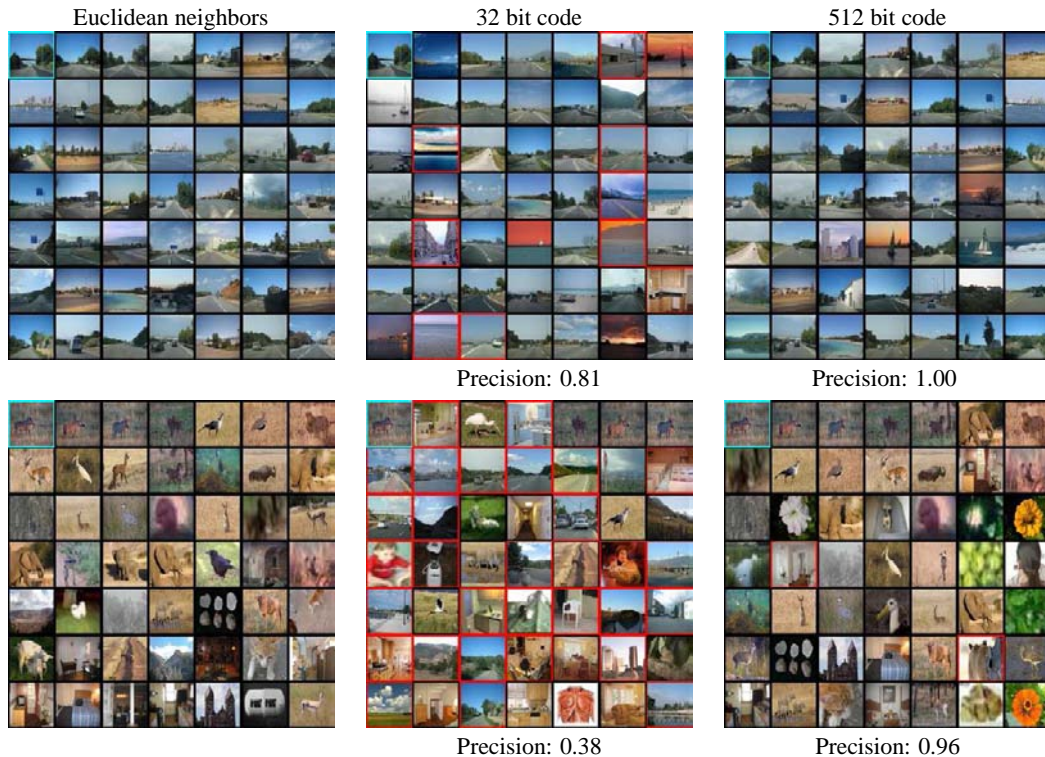

Figure 4: Examples of retrieval for two query images on the LabelMe database. The left column shows top 48 neighbors for each query according to Euclidean distance (the query image is in the top left of the collage). The middle (resp. right) column shows nearest neighbors according to normalized Hamming distance with a 32-bit (resp. 512-bit) code. The precision of retrieval is evaluated as the proportion of top Hamming neighbors that are also Euclidean neighbors within the "nominal" radius. Incorrectly retrieved images in the middle and right columns are shown with a red border. Best viewed in color.

at 0.2 recall, whereas spectral hashing only achieves about 0.5 precision at the same recall. Moreover, the performance of spectral hashing actually begins to decrease for code sizes above 256 bits. Finally, Figure 4 shows retrieval results for our method on a couple of representative query images.

In addition to being completely distribution-free and exhibiting more desirable behavior as a function of code size, our scheme has one more practical advantage. Unlike spectral hashing, we retain the kernel bandwidth $\gamma$ as a "free parameter," which gives us flexibility in terms of adapting to target neighborhood size, or setting a target Hamming distance for neighbors at a given Euclidean distance. This can be especially useful for making sure that a significant fraction of neighbors for each query are mapped to strings whose Hamming distance from the query is no greater than 2. This is a necesary condition for being able to use binary codes for hashing as opposed to brute-force search (although, as demonstrated in [11, 13], even brute-force search with binary codes can already be quite fast). To ensure high recall within a low Hamming radius, we can progressively increase the kernel bandwidth $\gamma$ as the code size increases, thus counteracting the increase in *unnormalized* Hamming distance that inevitably accompanies larger code sizes. Preliminary results (omitted for lack of space) show that this strategy can indeed increase recall for low Hamming radius while sacrificing some precision. In the future, we will evaluate this tradeoff more extensively, and test our method on datasets consisting of millions of data points. At present, our promising initial results, combined with our comprehensive theoretical analysis, convincingly demonstrate the potential usefulness of our scheme for large-scale indexing and search applications.

## Acknowledgments

This work was supported by NSF CAREER Award No. IIS 0845629.

## Footnotes

[1]We use the code made available by the authors of [15] at http://www.cs.huji.ac.il/˜yweiss/SpectralHashing/.

# References

[1] A. Andoni and P. Indyk. Near-optimal hashing algorithms for approximate nearest neighbor in high dimensions. *Commun. ACM*, 51(1):117–122, 2008.

[2] K. Clarkson. Nearest-neighbor searching and metric space dimensions. In *Nearest-Neighbor Methods for Learning and Vision: Theory and Practice*, pages 15–59. MIT Press, 2006.

[3] S. Dasgupta and Y. Freund. Random projection trees and low dimensional manifolds. In *STOC*, 2008.

[4] S. Dasgupta and A. Gupta. An elementary proof of a theorem of Johnson and Lindenstrauss. *Random Struct. Alg.*, 22(1):60–65, 2003.

[5] J. Heinonen. *Lectures on Analysis on Metric Spaces*. Springer, New York, 2001.

[6] P. Indyk and A. Naor. Nearest-neighbor-preserving embeddings. *ACM Trans. Algorithms*, 3(3):Art. 31, 2007.

[7] A. Oliva and A. Torralba. Modeling the shape of the scene: a holistic representation of the spatial envelope. *Int. J. Computer Vision*, 42(3):145–175, 2001.

[8] A. Rahimi and B. Recht. Random features for large-scale kernel machines. In *NIPS*, 2007.

[9] M. Reed and B. Simon. *Methods of Modern Mathematical Physics II: Fourier Analysis, Self-Adjointness*. Academic Press, 1975.

[10] B. Russell, A. Torralba, K. Murphy, and W. T. Freeman. LabelMe: a database and web-based tool for image annotation. *Int. J. Computer Vision*, 77:157–173, 2008.

[11] R. Salakhutdinov and G. Hinton. Semantic hashing. In *SIGIR Workshop on Inf. Retrieval and App. of Graphical Models*, 2007.

[12] B. Schölkopf and A. J. Smola. *Learning With Kernels*. MIT Press, 2002.

[13] A. Torralba, R. Fergus, and Y. Weiss. Small codes and large databases for recognition. In *CVPR*, 2008.

[14] A. W. van der Vaart and J. A. Wellner. *Weak Convergence and Empirical Processes*. Springer, 1996.

[15] Y. Weiss, A. Torralba, and R. Fergus. Spectral hashing. In *NIPS*, 2008.
